# Active learning for misspecified generalized linear models

**Francis R. Bach**
Centre de Morphologie Mathématique
Ecole des Mines de Paris
Fontainebleau, France
`francis.bach@mines.org`

## Abstract

Active learning refers to algorithmic frameworks aimed at selecting training data points in order to reduce the number of required training data points and/or improve the generalization performance of a learning method. In this paper, we present an asymptotic analysis of active learning for generalized linear models. Our analysis holds under the common practical situation of model misspecification, and is based on realistic assumptions regarding the nature of the sampling distributions, which are usually neither independent nor identical. We derive unbiased estimators of generalization performance, as well as estimators of expected reduction in generalization error after adding a new training data point, that allow us to optimize its sampling distribution through a convex optimization problem. Our analysis naturally leads to an algorithm for sequential active learning which is applicable for all tasks supported by generalized linear models (e.g., binary classification, multi-class classification, regression) and can be applied in non-linear settings through the use of Mercer kernels.

## 1  Introduction

The goal of active learning is to select training data points so that the number of required training data points for a given performance is smaller than the number which is required when randomly sampling those points. Active learning has emerged as a dynamic field of research in machine learning and statistics [1], from early works in optimal experimental design [2, 3], to recent theoretical results [4] and applications, in text retrieval [5], image retrieval [6] or bioinformatics [7].

Despite the numerous successful applications of active learning to reduce the number of required training data points, many authors have also reported cases where widely applied active learning heuristic schemes such as maximum uncertainty sampling perform worse than random selection [8, 9], casting doubt into the practical applicability of active learning: why would a practitioner use an active learning strategy that is not ensuring, unless the data satisfy possibly unrealistic and usually non verifiable assumptions, that it performs better than random? The objectives of this paper are (1) to provide a theoretical analysis of active learning with realistic assumptions and (2) to derive a principled algorithm for active learning with guaranteed consistency.

In this paper, we consider *generalized linear models* [10], which provide flexible and widely used tools for many supervised learning tasks (Section 2). Our analysis is based on asymptotic arguments, and follows previous asymptotic analysis of active learning [11, 12, 9, 13]; however, as shown in Section 4, we do not rely on correct model specification and assume that the data are not identically distributed and may not be independent. As shown in Section 5, our theoretical results naturally lead to convex optimization problems for selecting training data point in a sequential design. In Section 6, we present simulations on synthetic data, illustrating our algorithms and comparing them favorably to usual active learning schemes.

## 2    Generalized linear models

Given data $x \in \mathbb{R}^d$, and targets $y$ in a set $\mathcal{Y}$, we consider the problem of modeling the conditional probability $p(y|x)$ through a generalized linear model (GLIM) [10]. We assume that we are given an exponential family adapted to our prediction task, of the form $p(y|\eta) = \exp(\eta^\top T(y) - \psi(\eta))$, where $T(y)$ is a $k$-dimensional vector of sufficient statistics, $\eta \in \mathbb{R}^k$ is vector of natural parameters and $\psi(\eta)$ is the convex log-partition function. We then consider the generalized linear model defined as $p(y|x, \theta) = \exp(\mathrm{tr}(\theta^\top x T(y)^\top) - \psi(\theta^\top x))$, where $\theta \in \Theta \subset \mathbb{R}^{d \times k}$. The framework of GLIMs is general enough to accomodate many supervised learning tasks [10], in particular:

- Binary classification: the Bernoulli distribution leads to *logistic regression*, with $\mathcal{Y} = \{0, 1\}$, $T(y) = y$ and $\psi(\eta) = \log(1 + e^\eta)$.
- k-class classification: the multinomial distribution leads to *softmax regression*, with $\mathcal{Y} = \{y \in \{0, 1\}^k, \sum_{i=1}^k y_i = 1\}$, $T(y) = y$ and $\psi(\eta) = \log(\sum_{i=1}^k e^{\eta_i})$.
- Regression: the normal distribution leads to $\mathcal{Y} = \mathbb{R}$, $T(y) = (y, -\frac{1}{2} y^2)^\top \in \mathbb{R}^2$, and $\psi(\eta_1, \eta_2) = -\frac{1}{2} \log \eta_2 + \frac{1}{2} \log 2\pi + \frac{\eta_1^2}{2\eta_2}$. When both $\eta_1$ and $\eta_2$ depends linearly on $x$, we have an heteroscedastic model, while if $\eta_2$ is constant for all $x$, we obtain homoscedastic regression (constant noise variance).

**Maximum likelihood estimation**    We assume that we are given independent and identically distributed (i.i.d.) data sampled from the distribution $p_0(x, y) = p_0(x) p_0(y|x)$. The *maximum likelihood population estimator* $\theta_0$ is defined as the minimizer of the expectation under $p_0$ of the negative log-likelihood $\ell(y, x, \theta) = -\mathrm{tr}(\theta^\top x T(y)^\top) + \psi(\theta^\top x)$. The function $\ell(y, x, \theta)$ is convex in $\theta$ and by taking derivatives and using the classical relationship between the derivative of the log-partition and the expected sufficient statistics [10], the population maximum likelihood estimate is defined by:

$$E_{p_0(x,y)} \nabla \ell(y, x, \theta_0) = E_{p_0(x)} \left\{ x (E_{p(y|x,\theta_0)} T(y) - E_{p_0(y|x)} T(y))^\top \right\} = 0 \qquad (1)$$

Given i.i.d data $(x_i, y_i)$, $i = 1, \dots, n$, we use the penalized maximum likelihood estimator, which minimizes $\sum_{i=1}^n \ell(y_i, x_i, \theta) + \frac{1}{2} \lambda \mathrm{tr} \theta^\top \theta$. The minimization is performed by Newton's method [14].

**Model specification**    A GLIM is said *well-specified* is there exists a $\theta \in \mathbb{R}^{d \times k}$ such that for all $x \in \mathbb{R}^d$, $E_{p(y|x,\theta)} T(y) = E_{p_0(y|x)} T(y)$. A sufficient condition for correct specification is that there exist $\theta \in \mathbb{R}^{d \times k}$ such that for all $x \in \mathbb{R}^d$, $y \in \mathcal{Y}$, $p(y|x, \theta) = p_0(y|x)$. This condition is necessary for the Bernoulli and multinomial exponential family, but not for example for the normal distribution. In practice, the model is often misspecified and it is thus of importance to consider potential misspecification while deriving asymptotic expansions.

**Kernels**    The theoretical results of this paper mainly focus on generalized linear models; however, they can be readily generalized to non-linear settings by using Mercer kernels [15], for example leading to kernel logistic regression or kernel ridge regression. When the data are given by a kernel matrix, we can use the incomplete Cholesky decomposition [16] to find an approximate basis of the feature space on which the usual linear methods can be applied. Note that our asymptotic results do not hold when the number of parameters may grow with the data (which is the case for kernels such as the Gaussian kernel). However, our dimensionality reduction procedure uses a non-parametric method on the entire (usually large) training dataset and we then consider a finite dimensional problem on a much smaller sample. If the whole training dataset is large enough, then the dimension reduction procedure may be considered deterministic and our criteria may apply.

## 3    Active learning set-up

We consider the following "pool-based" active learning scenario: we have a large set of i.i.d. data points $x_i \in \mathbb{R}^d$, $i = 1, \dots, m$ sampled from $p_0(x)$. The goal of active learning is to select the points to label, i.e., the points for which the corresponding $y_i$ will be observed. We assume that given $x_i$, $i = 1, \dots, n$, the targets $y_i$, $i = 1, \dots, n$ are independent and sampled from the corresponding conditional distribution $p_0(y_i|x_i)$. This active learning set-up is well studied and appears naturally in many applications where the input distribution $p_0(x)$ is only known through i.i.d. samples [5, 17]. For alternative scenarii, where the density $p_0(x)$ is known, see e.g. [18, 19, 20].

More precisely, we assume that the points $x_i$ are selected sequentially, and we let denote $q_i(x_i|x_1, \ldots, x_{i-1})$ the sampling distribution of $x_i$ given the previously observed points. In situations where the data are not sampled from the testing distribution, it has proved advantageous to consider likelihood weighting techniques [13, 19], and we thus consider weights $w_i = w_i(x_i|x_1, \ldots, x_{i-1})$. We let $\hat{\theta}_n$ denote the weighted penalized ML estimator, defined as the minimum with respect to $\theta$ of

$$\sum_{i=1}^n w_i \ell(y_i, x_i, \theta) + \frac{\lambda}{2} \text{tr} \theta^\top \theta. \tag{2}$$

In this paper, we work with two different assumptions regarding the sequential sampling distributions: (1) the variables $x_i$ are independent, i.e., $q_i(x_i|x_1, \ldots, x_{i-1}) = q_i(x_i)$, (2) the variable $x_i$ depends on $x_1, \ldots, x_{i-1}$ only through the current empirical ML estimator $\hat{\theta}_i$, i.e., $q_i(x_i|x_1, \ldots, x_{i-1}) = q(x_i|\hat{\theta}_i)$, where $q(x_i|\theta)$ is a pre-specified sampling distribution. The first assumption is not realistic, but readily leads to asymptotic expansions. The second assumption is more realistic, as most of the heuristic schemes for sequential active learning satisfy this assumption. It turns out that under certain assumption, the asymptotic expansions of the expected generalization performance for both sets of assumptions are identical.

## 4  Asymptotic expansions

In this section, we derive the asymptotic expansions that will lead to active learning algorithms in Section 5. Throughout this section, we assume that $p_0(x)$ has a compact support $K$ and has a twice differentiable density with respect to the Lebesgue measure, and that all sampling distributions have a compact support included in the one of $p_0(x)$ and have twice differentiable densities.

We first make the assumption that the variables $x_i$ are *independent*, i.e., we have sampling distributions $q_i(x_i)$ and weights $w_i(x_i)$, both measurable, and such that $w_i(x_i) > 0$ for all $x_i \in K$. In Section 4.4, we extend some of our results to the dependent case.

### 4.1  Bias and variance of ML estimator

The following proposition is a simple extension to non identically distributed observations, of classical results on maximum likelihood for misspecified generalized linear models [21, 13]. We let $E_{\mathcal{D}}$ and $\text{var}_{\mathcal{D}}$ denote the expectation and variance with respect to the data $\mathcal{D} = \{(x_i, y_i),\ i = 1, \ldots, n\}$.

**Proposition 1** *We let $\theta_n$ denote the minimizer of $\sum_{i=1}^n E_{q_i(x_i)p_0(y_i|x_i)} w_i(x_i) \ell(y_i, x_i, \theta)$. If (a) the weight functions $w_n$ and the sampling densities $q_n$ are pointwise strictly positive and such that $w_n(x)q_n(x)$ converges in the $L^\infty$-norm, and (b) $E_{q_n(x)} w_n^2(x)$ is bounded , then $\hat{\theta}_n - \theta_n$ converges to zero in probability and we have*

$$E_{\mathcal{D}}\hat{\theta}_n = \theta_n + O(n^{-1}) \text{ and } \text{var}_{\mathcal{D}}\hat{\theta}_n = \frac{1}{n}J_n^{-1}I_n J_n^{-1} + O(n^{-2}) \tag{3}$$

*where $J_n = \frac{1}{n}\sum_{i=1}^n E_{q_i(x)} w_i(x) \nabla^2 \ell(x, \theta_n)$ can be consistently estimated by $\hat{J}_n = \frac{1}{n}\sum_{i=1}^n w_i h_i$ and $I_n = \frac{1}{n}\sum_{i=1}^n E_{q_i(x)p_0(y|x)} w_i(x)^2 \nabla \ell(y, x, \theta_n) \nabla \ell(y, x, \theta_n)^\top$ can be consistently estimated by $\hat{I}_n = \frac{1}{n}\sum_{i=1}^n w_i^2 g_i g_i^\top$, where $g_i = \nabla \ell(y_i, x_i, \hat{\theta}_n)$ and $h_i = \nabla^2 \ell(x_i, \hat{\theta}_n)$.*

From Proposition 1, it is worth noting that in general $\theta_n$ will not converge to the population maximum likelihood estimate $\theta_0$, i.e., using a different sampling distribution than $p_0(x)$ may introduce a non asymptotically vanishing bias in estimating $\theta_0$. Thus, active learning requires to ensure that (a) our estimators have a low bias and variance in estimating $\theta_n$, and (b) that $\theta_n$ does actually converge to $\theta_0$. This double objective is taken care of by our estimates of generalization performance in Propositions 2 and 3.

There are two situations, however, where $\theta_n$ is equal to $\theta_0$. First, if the model is well specified, then whatever the sampling distributions are, $\theta_n$ is the population ML estimate (which is a simple consequence of the fact that $E_{p(y|x,\theta_0)} T(y) = E_{p_0(y|x)} T(y)$, for all $x$, implies that, for all $q(x)$, $E_{q(x)p_0(y|x)} \nabla \ell(y, x, \theta) = E_{q(x)} \left\{ x(E_{p(y|x,\theta_0)} T(y) - E_{p_0(y|x)} T(y))^\top \right\} = 0$).

Second, When $w_n(x) = p_0(x)/q_n(x)$, then $\theta_n$ is also equal to $\theta_0$, and we refer to this weighting scheme as the unbiased reweighting scheme, which was used by [19] in the context of active learning. We refer to the weights $w_n^u = p_0(x_n)/q_n(x_n)$ as the *importance* weights. Note however, that

restricting ourselves to such unbiased estimators, as done in [19] might not be optimal because they may lead to higher variance [13], in particular due to the potential high variance of the importance weights (see simulations in Section 6).

## 4.2 Expected generalization performance

We let $L^u(\theta) = E_{p_0(x)p_0(y|x)}\ell(y, x, \theta)$ denote the generalization performance[1] of the parameter $\theta$. We now provide an unbiased estimator of the expected generalization error of $\hat{\theta}_n$, which generalized the Akaike information criterion [22] (for a proof, see [23]):

**Proposition 2** *In addition to the assumptions of Proposition 1, we assume that $E_{q_n(x)}\left(p_0(x)/q_n(x)\right)^2$ is bounded. Let*

$$\widehat{G} = \frac{1}{n}\sum_{i=1}^{n} w_i^u \ell(y_i, x_i, \hat{\theta}_n) + \frac{1}{n}\left(\frac{1}{n}\sum_{i=1}^{n} w_i^u w_i g_i^\top (\hat{J}_n)^{-1} g_i\right), \qquad (4)$$

*where $w_i^u = p_0(x_i)/q_i(x_i)$. $\widehat{G}$ is an asymptotically unbiased estimator of $E_{\mathcal{D}}L^u(\hat{\theta}_n)$, i.e., $E_{\mathcal{D}}\widehat{G} = E_{\mathcal{D}}L^u(\hat{\theta}_n) + O(n^{-2})$.*

The criterion $\widehat{G}$ is a sum of two terms: the second term corresponds to a variance term and will converge to zero in probability at rate $O(n^{-1})$; the first term, however, which corresponds to a selection bias induced by a specific choice of sampling distributions, will not always converge to the minimum possible value $L^u(\theta_0)$. Thus, in order to ensure that our active learning method are consistent, we have to ensure that this first term is going to its minimum value. One simple way to achieve this is to always optimize our weights so that the estimate $\widehat{G}$ is smaller than the estimate for the unbiased reweighting scheme (see Section 5).

## 4.3 Expected performance gain

We now look at the following situation: we are given the first $n$ data points $(x_i, y_i)$ and the current estimate $\hat{\theta}_n$, the gradients $g_i = \nabla\ell(y_i, x_i, \hat{\theta}_n)$, the Hessians $h_i = \nabla^2\ell(x_i, \hat{\theta}_n)$ and the third derivatives $T_i = \nabla^3\ell(x_i, \hat{\theta}_n)$, we consider the following criterion, which depends on the sampling distributions and weights of the $(n+1)$-th point:

$$\widehat{H}(q_{n+1}, w_{n+1}|\alpha, \beta) = \frac{1}{n^3}\sum_{i=1}^{n}\alpha_i w_i^u w_{n+1}(x_i)\frac{q_{n+1}(x_i)}{p_0(x_i)} + \sum_{i=1}^{n}\beta_i w_i^u w_{n+1}(x_i)^2 \frac{q_{n+1}(x_i)}{p_0(x_i)} \quad (5)$$

$$\text{where} \quad \alpha_i = -(n+1)n\tilde{g}_i^\top \hat{J}_n A - w_i w_i^u \tilde{g}_i^\top h_i \tilde{g}_i + w_i^u \tilde{g}_i^\top \hat{J}_n \tilde{g}_i - 2\tilde{g}_i^\top B$$
$$-w_i \tilde{g}_i^\top \hat{J}_n^u \tilde{g}_i + T_i[\tilde{g}_i, C] - 2w_i \tilde{g}_i^\top h_i A + T_i[A, \tilde{g}_i, \tilde{g}_i] \quad (6)$$

$$\beta_i = \frac{1}{2}\tilde{g}_i^\top \hat{J}_n^u \tilde{g}_i + A^\top h_i \tilde{g}_i \quad (7)$$

with $\tilde{g}_i = \hat{J}_n^{-1}g_i$, $A = \hat{J}_n^{-1}\frac{1}{n}\sum_{i=1}^{n}w_i^u g_i$, $B = \sum_{i=1}^{n}w_i^u w_i h_i \tilde{g}_i$, $C = \sum_{i=1}^{n}w_i w_i^u \tilde{g}_i \tilde{g}_i^\top$, $\hat{J}_n^u = \frac{1}{n}\sum_{i=1}^{n}w_i^u h_i$.

The following proposition shows that $\widehat{H}(q_{n+1}, w_{n+1}|\alpha, \beta)$ is an estimate of the expected performance gain of choosing a point $x_{n+1}$ according to distribution $q_{n+1}$ and weight $w_{n+1}$ (and marginalizing over $y_{n+1}$) and may be used as an objective function for learning the distributions $q_{n+1}, w_{n+1}$ (for a proof, see [23]). In Section 5, we show that if the distributions and weights are properly parameterized, this leads to a convex optimization problem.

**Proposition 3** *We assume that $E_{q_n(x)}w_n^2(x)$ and $E_{q_n(x)}\left(p_0(x)/q_n(x)\right)^2$ are bounded. We let denote $\hat{\theta}_n$ denote the weighted ML estimator obtained from the first $n$ points, and $\hat{\theta}_{n+1}$ the one-step estimator obtained from the first $n+1$ points, i.e., $\hat{\theta}_{n+1}$ is obtained by one Newton step from $\hat{\theta}_n$ [24]; then the criterion defined in Eq. (5) is such that $E_{\mathcal{D}}\widehat{H}(q_{n+1}, w_{n+1}) = E_{\mathcal{D}}L^u(\hat{\theta}_n) - E_{\mathcal{D}}L^u(\hat{\theta}_{n+1}) + O(n^{-3})$, where $E_{\mathcal{D}}$ denotes the expectation with respect to the first $n+1$ data points and their labels. Moreover, for $n$ large enough, all values of $\beta_i$ are positive.*

Note that many of the terms in Eq. (6) and Eq. (7) are dedicated to weighting schemes for the first $n$ points other than the unbiased reweighting scheme. For the unbiased reweighting scheme where $w_i = w_i^u$, for $i = 1, \ldots, n$, then $A = 0$ and the equations may be simplified.

## 4.4 Dependent observations

In this section, we show that under a certain form of weak dependence between the data points $x_i$, $i = 1, \ldots, n$, then the results presented in Propositions 1 and 2 still hold. For simplicity and brevity, we restrict ourselves to the unbiased reweighting scheme, i.e., $w_n(x_n|x_1, \ldots, x_{n-1}) = p_0(x_n)/q_n(x_n|x_1, \ldots, x_{n-1})$ for all $n$, and we assume that those weights are uniformly bounded away from zero and infinity. In addition, we only prove our result in the well-specified case, which leads to a simpler argument for the consistency of the estimator.

Many sequential active learning schemes select a training data point with a distribution or criterion that depends on the estimate so far (see Section 6 for details). We thus assume that the sampling distribution $q_n$ is of the form $q(x_n|\hat{\theta}_n)$, where $q(x|\theta)$ is a fixed set of smooth parameterized densities.

**Proposition 4** *(for a proof, see [23]) Let*

$$\widehat{G} = \tfrac{1}{n} \sum_{i=1}^n w_i \ell(y_i, x_i, \hat{\theta}_n) + \tfrac{1}{n} \left( \tfrac{1}{n} \sum_{i=1}^n w_i^2 g_i^\top (\hat{J}_n)^{-1} g_i \right), \tag{8}$$

*where $w_i = w_i^u = p_0(x_i)/q(x_i|\hat{\theta}_i)$. $\widehat{G}$ is an asymptotically unbiased estimator of $E_{\mathcal{D}} L^u(\hat{\theta}_n)$, i.e., $E_{\mathcal{D}} \widehat{G} = E_{\mathcal{D}} L^u(\hat{\theta}_n) + O(\log(n) n^{-2})$.*

The estimator is the same as in Proposition 2. The effect of the dependence is asymptotically negligible and only impacts the result with the presence of an additional $\log(n)$ term. In the algorithms presented in Section 5, the distribution $q_n$ is obtained as the solution of a convex optimization problem, and thus the previous theorem does not readily apply. However, when $n$ gets large, $q_n$ depends on the previous data points only through the first two derivatives of the objective function of the convex problem, which are empirical averages of certain functions of all currently observed data points; we are currently working out a generalization of Proposition 4 that allows the dependence on certain empirical moments and potential misspecification.

## 5 Algorithms

In Section 4, we have derived a criterion $\widehat{H}$ in Eq. (5) that enables to optimize the sampling density of the $(n + 1)$-th point, and an estimate $\widehat{G}$ in Eq. (4) and Eq. (8) of the generalization error. Our algorithms are composed of the following three ingredients:

1. Those criteria assume that the variance of the importance weights $w_n^u = p_0(x_n)/q_n(x_n)$ is controlled. In order to make sure that those results apply, our algorithms will ensure that this condition is met.

2. The sampling density $q_{n+1}$ will be obtained by minimizing $\widehat{H}(w_{n+1}, q_{n+1}|\alpha, \beta)$ for a certain parameterization of $q_{n+1}$ and $w_{n+1}$. It turns out that those minimization problems are *convex*, and can thus be efficiently solved, without local minima.

3. Once a new sample has been selected, and its label observed, Proposition 4 is used in a way similar to [13], in order to search for the best mixture between the current weights $(w_i)$ and the importance weights $(w_i^u)$, i.e., we look at weights of the form $w_i^\gamma (w_i^u)^{1-\gamma}$ and perform a grid search on $\gamma$ to find $\gamma$ such that $\widehat{G}$ in Eq. (4) is minimum.

The main interest of the first and third points is that we obtain a final estimator of $\theta_0$ which is at least provably consistent: indeed, although our criteria are obtained from an assumption of independence, the generalization performance result also holds for "weakly" dependent observations and thus ensures the consistency of our approach. Thus, as opposed to most previous active learning heuristics, our estimator will always converge (in probability) to the ML estimator. In Section 6, we show empirically that usual heuristic schemes do not share this property.

**Convex optimization problem** We assume that we have a fixed set of candidate distributions $s_k(x)$ of the form $s_k(x) = p_0(x)r_k(x)$. Note that the multiplicative form of our candidate distri-

butions allows efficient sampling from a pool of samples of $p_0$. We look at distributions $q_{n+1}(x)$ with mixture density of the form $s(x|\eta) = \sum_k \eta_k s_k(x) = p_0(x)r(x)$, where the weights $\eta$ are non-negative and sum to one. The criterion $\widehat{H}(q_{n+1}, w_{n+1}|\alpha, \beta)$ in Eq. (5) is thus a function $H(\eta|\alpha, \beta)$ of $\eta$. We consider two weighting schemes: (a) one with all weights equal to one (unit weighting scheme) which leads to $H_0(\eta|\alpha, \beta)$, and (b) the unbiased reweighting scheme, where $w_{n+1}(x) = p_0(x)/q_{n+1}(x)$, which leads to $H_1(\eta|\alpha, \beta)$. We have

$$H_0(\eta|\alpha, \beta) = \frac{1}{n^3} \sum_k \eta_k \left( \sum_{i=1}^n (\alpha_i + \beta_i) w_i^u s_k(x_i) \right), \quad (9)$$

$$H_1(\eta|\alpha, \beta) = \frac{1}{n^3} \sum_{i=1}^n \alpha_i w_i^u + \sum_{i=1}^n \frac{\beta_i w_i^u}{\sum_k \eta_k s_k(x_i)}. \quad (10)$$

The function $H_0(\eta)$ is linear in $\eta$, while the function $H_1(\eta)$ is the sum of a constant and positive inverse functions, and is thus convex [14].

Unless natural candidate distributions $s_k(x)$ can be defined for the active learning problem, we use the set of distributions obtained as follows: we perform K-means clustering with a large number $p$ of clusters (e.g., 100 or 200), and then consider functions $r_k(x)$ of the form $r_k(x) = \frac{1}{Z_k} e^{-\alpha_k \|x - \mu_k\|^2}$, where $\alpha_k$ is one element of a finite given set of parameters, and $\mu_k$ is one of the $p$ centroids $y_1, \ldots, y_p$, obtained from K-means. We let $\tilde{w}_i$ denote the number of data points assigned to the centroid $y_i$. We normalize by $Z_k = \sum_{i=1}^p \tilde{w}_i e^{-\alpha_k \|y_i - \mu_k\|^2} / \sum_{i=1}^p \tilde{w}_i$. We thus obtained $O(p)$ candidate distributions $r_k(x)$, which, if $p$ is large enough, provides a flexible yet tractable set of mixture distributions.

One additional element is the constraint on the variance of the importance weights. The variance of $w_{n+1}^u$ can be estimated as $\operatorname{var} w_{n+1}^u = \sum_{i=1}^m \frac{\tilde{w}_i}{r(x_i)} - 1 = \sum_{i=1}^m \frac{\tilde{w}_i}{\sum_k \eta_k r_k(x_i)} - 1 = V(\eta)$, which is convex in $\eta$. Thus constraining the variance of the new weights leads to a convex optimization problem, with convex objective and convex constraints, which can be solved efficiently by the log-barrier method [14], with cubic complexity in the number of candidate distributions.

**Algorithms** We have three versions of our algorithm, one with unit weights (referred to as "no weight") which optimizes $H_0(\eta|\alpha, \beta)$ at each iteration, one with the unbiased reweighting scheme, which optimizes $H_1(\eta|\alpha, \beta)$ (referred to as "unbiased") and one which does both and chooses the best one, as measured by $\widehat{H}$ (referred to as "full"): in the initialization phase, K-means is run to generate candidate distributions that will be used throughout the sampling of new points. Then, in order to select the new training data point $x_{n+1}$, the scores $\alpha$ and $\beta$ are computed from Eq. (6) and Eq. (7), then the appropriate cost function, $H_0(\eta|\alpha, \beta)$, $H_1(\eta|\alpha, \beta)$ (or both) is minimized and once $\eta$ is obtained, we sample $x_{n+1}$ from the corresponding distribution, and compute the weights $w_{n+1}$ and $w_{n+1}^u$. As described earlier, we then find $\gamma$ such that $\widehat{G}((w_i^\gamma (w_i^u)^{1-\gamma})_i)$ in Eq. (4) is minimized and update weights accordingly.

**Regularization parameter** In the active learning set-up, the number of samples used for learning varies a lot. It is thus not possible to use a constant regularization parameter. We thus learn it by cross-validation every 10 new samples.

## 6 Simulation experiments

In this section, we present simulation experiments on synthetic examples (sampled from Gaussian mixtures in two dimensions), for the task of binary and 3-class classification. We compare our algorithms to the following three active learning frameworks. In the *maximum uncertainty* framework (referred to as "maxunc"), the next training data point is selected such that the entropy of $p(y|x, \hat{\theta}_n)$ is maximal [17]. In the *maximum variance reduction* framework [25, 9] (referred to as "varred"), the next point is selected so that the variance of the resulting estimator has the lowest determinant, which is equivalent to finding $x$ such that $\operatorname{tr}\nabla(x, \hat{\theta}_n)\hat{J}_n^{-1}$ is minimum. Note that this criterion has theoretical justification under correct model specification. In the *minimum prediction error* framework (referred to as "minpred"), the next point is selected so that it reduces the most the expected log-loss, with the current model as an estimate of the unknown conditional probability $p_0(y|x)$ [5, 8].

**Sampling densities** In Figure 1, we look at the limit selected sampling densities, i.e., we assume that a large number of points has been sampled, and we look at the criterion $\widehat{H}$ in Eq. (5). We show the density obtained from the unbiased reweighting scheme (middle of Figure 1), as well as

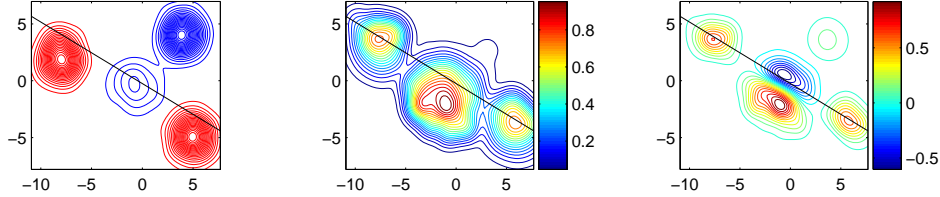

Figure 1: Proposal distributions: (Left) density $p_0(x)$ with the two different classes (red and blue), (Middle) best density with unbiased reweighting, (Right) function $\gamma(x)$ such that $\widehat{H}(q_{n+1}(x), 1) = \int \gamma(x)q_{n+1}(x)dx$ (see text for details).

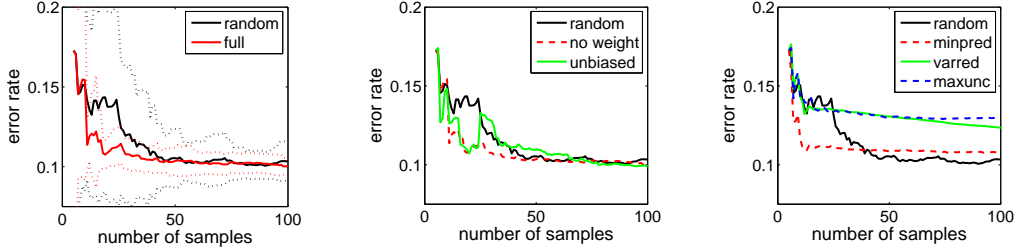

Figure 2: Error rates vs. number of samples averaged over 10 replications sampled from same distribution as in Figure 1: (Left) random sampling and active learning "full", with standard deviations, (Middle) Comparison of the two schemes "unbiased" and "no weight", (Right) Comparison with other methods.

the function $\gamma(x)$ (right of Figure 1) such that, for the unit weighting scheme, $\widehat{H}(q_{n+1}(x), 1) = \int \gamma(x)q_{n+1}(x)dx$. In this framework, minimizing the cost without any constraint leads to a Dirac at the maximum of $\gamma(x)$, while minimizing with a constraint on the variance of the corresponding importance weights will select point with high values of $\gamma(x)$. We also show the line $\theta_0^\top x = 0$. From Figure 1, we see that (a) the unit weighting scheme tends to be more selective (i.e., finer grain) than the unbiased scheme, and (b) that the mode of the optimal densities are close to the maximum uncertainty hyperplane but some parts of this hyperplane are in fact leading to negative cost gains (e.g., the part of the hyperplane crossing the central blob), hinting at the bad potential behavior of the maximum uncertainty framework.

**Comparison with other algorithms**     In Figure 2 and Figure 1, we compare the performance of our active learning algorithms. In the left of Figure 2, we see that our active learning framework does perform better on average but also leads to smaller variance. In the middle of Figure 2, we compare the two schemes "no weight" and "unbiased", showing the superiority of the unit weighting scheme and the significance of our asymptotic results in Proposition 2 and 3 which extend the unbiased framework of [13]. In the right of Figure 2 and in Figure 3, we compare with the other usual heuristic schemes: our "full" algorithm outperforms other schemes; moreover, in those experiments, the other schemes do perform worse than random sampling and converge to the wrong estimator, a bad situation that our algorithms provably avoid.

# 7  Conclusion

We have presented a theoretical asymptotic analysis of active learning for generalized linear models, under realistic sampling assumptions. From this analysis, we obtain convex criteria which can be optimized to provide algorithms for online optimization of the sampling distributions. This work naturally leads to several extensions. First, our framework is not limited to generalized linear models, but can be readily extended to any convex differentiable $M$-estimators [24]. Second, it seems advantageous to combine our active learning analysis with semi-supervised learning frameworks, in particular ones based on data-dependent regularization [26]. Finally, we are currently investigating applications to large scale image retrieval tasks, where unlabelled data are abundant but labelled data are scarce.

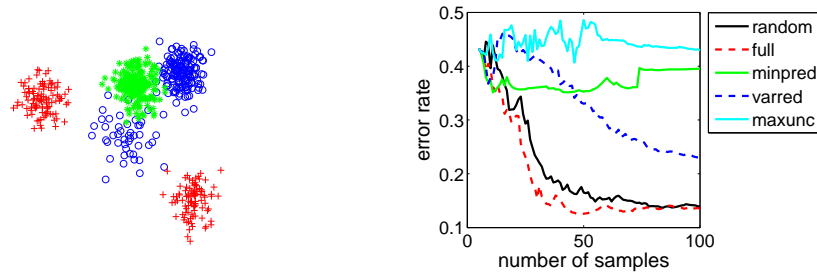

Figure 3: Error rates vs. number of samples averaged over 10 replications for 3 classes: (left) data, (right) comparisons of methods.

## Footnotes

[1]In this paper, we use the negative log-likelihood as a measure of performance, which allows simple asymptotic expansions, and the focus of the paper is about the differences between testing and training sampling distributions. The study of potentially different costs for testing and training is beyond the scope of this paper.

# References

[1] D. A. Cohn, Z. Ghahramani, and M. I. Jordan. Active learning with statistical models. *J. Art. Intel. Res.*, 4:129–145, 1996.

[2] V. V. Fedorov. *Theory of optimal experiments*. Academic Press, 1972.

[3] P. Chaudhuri and P. A. Mykland. On efficient designing of nonlinear experiments. *Stat. Sin.*, 5:421–440, 1995.

[4] S. Dasgupta. Coarse sample complexity bounds for active learning. In *Adv. NIPS 18*, 2006.

[5] N. Roy and A. McCallum. Toward optimal active learning through sampling estimation of error reduction. In *Proc. ICML*, 2001.

[6] S. Tong and E. Chang. Support vector machine active learning for image retrieval. In *Proc. ACM Multimedia*, 2001.

[7] M. Warmuth, G. Rätsch, M. Mathieson, J. Liao, and C. Lemmen. Active learning in the drug discovery process. In *Adv. NIPS 14*, 2002.

[8] X. Zhu, J. Lafferty, and Z. Ghahramani. Combining active learning and semi-supervised learning using Gaussian fields and harmonic functions. In *Proc. ICML*, 2003.

[9] A I. Schein. *Active Learning for Logistic Regression*. Ph.D. diss., U. Penn., 2005. CIS Dpt.

[10] P. McCullagh and J. A. Nelder. *Generalized Linear Models*. Chapman and Hall, 1989.

[11] T. Zhang and F. J. Oles. A probability analysis on the value of unlabeled data for classification problems. In *Proc. ICML*, 2000.

[12] O. Chapelle. Active learning for parzen window classifier. In *Proc. AISTATS*, 2005.

[13] H. Shimodaira. Improving predictive inference under covariate shift by weighting the log-likelihood function. *J. Stat. Plan. Inf.*, 90:227–244, 2000.

[14] S. Boyd and L. Vandenberghe. *Convex Optimization*. Cambridge Univ. Press, 2003.

[15] J. Shawe-Taylor and N. Cristianini. *Kernel Methods for Pattern Analysis*. Cambridge Univ. Press, 2004.

[16] S. Fine and K. Scheinberg. Efficient SVM training using low-rank kernel representations. *J. Mach. Learn. Res.*, 2:243–264, 2001.

[17] S. Tong and D. Koller. Support vector machine active learning with applications to text classification. In *Proc. ICML*, 2000.

[18] K. Fukumizu. Active learning in multilayer perceptrons. In *Adv. NIPS 8*, 1996.

[19] T. Kanamori and H. Shimodaira. Active learning algorithm using the maximum weighted log-likelihood estimator. *J. Stat. Plan. Inf.*, 116:149–162, 2003.

[20] T. Kanamori. Statistical asymptotic theory of active learning. *Ann. Inst. Stat. Math.*, 54(3):459–475, 2002.

[21] H. White. Maximum likelihood estimation of misspecified models. *Econometrica*, 50(1):1–26, 1982.

[22] H. Akaike. A new look at statistical model identification. *IEEE Trans. Aut. Cont.*, 19:716–722, 1974.

[23] F. R. Bach. Active learning for misspecified generalized linear models. Technical Report N15/06/MM, Ecole des Mines de Paris, 2006.

[24] A. W. Van der Vaart. *Asymptotic Statistics*. Cambridge Univ. Press, 1998.

[25] D. MacKay. Information-based objective functions for active data selection. *Neural Computation*, 4(4):590–604, 1992.

[26] Y. Bengio and Y Grandvalet. Semi-supervised learning by entropy minimization. In *Adv. NIPS 17*, 2005.
